# General-purpose localization of textured image regions

**Ruth Rosenholtz***
Xerox PARC
3333 Coyote Hill Rd.
Palo Alto, CA 94304

## Abstract

We suggest a working definition of texture: Texture is stuff that is more compactly represented by its statistics than by specifying the configuration of its parts. This definition suggests that to find texture we look for outliers to the local statistics, and label as texture the regions with no outliers. We present a method, based upon this idea, for labeling points in natural scenes as belonging to texture regions, while simultaneously allowing us to label low-level, bottom-up cues for visual attention. This method is based upon recent psychophysics results on processing of texture and popout.

## 1 WHAT IS TEXTURE, AND WHY DO WE WANT TO FIND IT?

In a number of problems in computer vision and image processing, one must distinguish between image regions that correspond to objects and those which correspond to texture, and perform different processing depending upon the type of region. Current computer vision algorithms assume one magically knows this region labeling. But what is texture? We have the notion that texture involves a pattern that is somehow homogeneous, or in which signal changes are "too complex" to describe, so that aggregate properties must be used instead (Saund, 1998). There is by no means a firm division between texture and objects; rather, the characterization often depends upon the scale of interest (Saund, 1998).

Ideally the definition of texture should probably depend upon the application. We investigate a definition that we believe will be of fairly general utility: Texture is stuff that seems to belong to the local statistics. We propose extracting several texture features, at several different scales, and labeling as texture those regions whose feature values are likely to have come from the local distribution.

Outliers to the local statistics tend to draw our attention (Rosenholtz, 1997, 1998). The phenomenon is often referred to as "popout." Thus while labeling (locally) statistically homogeneous regions as texture, we can simultaneously highlight salient outliers to the local statistics. Our revised definition is that texture is the absence of popout.

In Section 2, we discuss previous work in both human perception and in finding texture and regions of interest in an image. In Section 3, we describe our method. We present and discuss results on a number of real images in Section 4.

## 2   PREVIOUS WORK

See (Wolfe, 1998) for a review of the visual search literature. Popout is typically studied using simple displays, in which an experimental subject searches for the unusual, target item, among the other, distractor items. One typically attempts to judge the "saliency," or degree to which the target pops out, by studying the efficiency of search for that item. Typically popout is modeled by a relatively low-level operator, which operates independently on a number of basic features of the image, including orientation, contrast/color, depth, and motion. In this paper, we look only at the features of contrast and orientation.

Within the image-processing field, much of the work in finding texture has defined as texture any region with a high luminance variance, e.g. Vaisey & Gersho (1992). Unfortunately, the luminance variance in a region containing an edge can be as high as that in a textured region. Won & Park (1997) use model fitting to detect image blocks containing an edge, and then label blocks with high variance as containing texture.

Recently, several computer vision researchers have also tackled this problem. Leung & Malik (1996) found regions of completely deterministic texture. Other researchers have used the definition that if the luminance goes up and then down again (or vice versa) it's texture (Forsyth et al, 1996). However, this method will treat lines as if they were texture. Also, with no notion of similarity within a texture (also lacking in the image-processing work), one would mark a "fault" in a texture as belonging to that texture. This would be unacceptable for a texture synthesis application, in which a routine that tried to synthesize such a texture would most likely fail to reproduce the (highly visible) fault. More recently, Shi and Malik (1998) presented a method for segmenting images based upon texture features. Their method performs extremely well at the segmentation task, dividing an image into regions with internal similarity that is high compared to the similarity across regions. However, it is difficult to compare with their results, since they do not explicitly label a subset of the resulting regions as texture. Furthermore, this method may also tend to mark a "fault" in a texture as belonging to that texture. This is both because the method is biased against separating out small regions, and because the grouping of a patch with one region depends as much upon the difference between that patch and other regions as it does upon the similarity between the patch and the given region.

Very little computer vision work has been done on attentional cues. Milanese et al (1993) found salient image regions using both top-down information and a bottom-up "conspicuity" operator, which marks a local region as more salient the greater the

difference between a local feature value and the mean feature value in the surrounding region. However, for the same difference in means, a local region is less salient when there is a greater variance in the feature values in the surrounding region (Duncan & Humphreys, 1989; Rosenholtz, 1997). We use as our saliency measure a test for outliers to the local distribution. This captures, in many cases, the dependence of saliency on difference between a given feature value and the local mean, relative to the local standard deviation. We will discuss our saliency measure in greater detail in the following section.

## 3  FINDING TEXTURE AND REGIONS OF INTEREST

We compute multiresolution feature maps for orientation and contrast, and then look for outliers in the local orientation and contrast statistics. We do this by first creating a 3-level Gaussian pyramid representation of the image. To extract contrast, we filter the pyramid with a difference of circularly symmetric Gaussians. The response of these filters will oscillate, even in a region with constant-contrast texture (e.g. a sinewave pattern). We approximate a computation of the maximum response of these filters over a small region by first squaring the filter responses, and then filtering the contrast energy with an appropriate Gaussian. Finally, we threshold the contrast to eliminate low-contrast regions ("flat" texture). These thresholds (one for each scale) were set by examining the visibility of sinewave patterns of various spatial frequencies.

We compute orientation in a simple and biologically plausible way, using Bergen & Landy's (1991) "back pocket model" for low-level computations:

1. Filter the pyramid with horizontal, vertical, and ±45° oriented Gaussian second derivatives.

2. Compute opponent energy by squaring the filter outputs, pooling them over a region 4 times the scale of the second derivative filters, and subtracting the vertical from the horizontal response and the +45° from the -45° response.

3. Normalize the opponent energy at each scale by dividing by the total energy in the 4 orientation energy bands at that scale.

The result is two images at each scale of the pyramid. To a good approximation, in regions which are strongly oriented, these images represent $k\cos(2\theta)$ and $k\sin(2\theta)$, where $\theta$ is the local orientation at that scale, and $k$ is a value between 0 and 1 which is related to the local orientation specificity. Orientation estimates from points with low specificity tend to be very noisy. In images of white noise, 80% of the estimates of $k$ fall below 0.5, therefore with 80% confidence, an orientation specificity of $k>0.5$ did not occur due to chance. We use this value to threshold out orientation estimates with low "orientedness."

We then estimate $D$, the local feature distribution, for each feature and scale, using the method of Parzen windows. The blurring of the distribution estimate by the Parzen window mimics uncertainty in estimates of feature values by the visual system. We collect statistics over a local *integration* region. For texture processing, the size of this region is independent of viewing distance, and is roughly $10S$ in diameter, where $S$ is the support of the Gaussian $2^{nd}$ derivative filters used to extract the texture features (Kingdom & Keeble, 1997; Kingdom et al, 1995).

We next compute a non-parametric measure of saliency:

$$\text{saliency} = -\log\left(\frac{P(v\,|\,D)}{\max_{x} P(x\,|\,D)}\right) \tag{1}$$

Note that if $D$ were Gaussian $N(\mu, \sigma^2)$, this simplifies to

$$\frac{(x - \mu)^2}{2\sigma^2} \qquad (2)$$

which should be compared to the standard parametric test for outliers, which uses the measure $(x - \mu)/\sigma$. Our saliency measure is essentially a more general, non-parametric form of this measure (i.e. it does not assume a Gaussian distribution).

Points with saliency less than 0.5 are labeled as candidate texture points. If $D$ were Gaussian, this would correspond to feature estimates within one standard deviation of the mean. Points with saliency greater than 3.1 are labeled as candidates for bottom-up attentional cues. If $D$ were Gaussian, this would correspond to feature estimates more than $2.5\sigma$ from the mean, a standard parametric test for outliers. One could, of course, keep the raw saliency values, as a measure of the likelihood that a region contained texture, rather than setting a hard threshold. We use a hard threshold in our examples to better display the results. Both the texture images and the region of interest images are median-filtered to remove extraneous points.

## 4  EXPERIMENTAL RESULTS

Figure 3 shows several example images. Figures 2, 3, and 4 show texture found at each scale of processing. The striped and checkered patterns represent oriented and homogeneous contrast texture, respectively. The absence of an image in any of these figures means that no texture of the given type was found in that image at the given scale. Note that we perform no segmentation of one texture from another.

For the building image, the algorithm labeled bricks and window panes as fine-scale texture, and windows and shutters as coarser-scale texture. The leopard skin and low-frequency stripes in the lower right corner of the leopard image were correctly labeled as texture. In the desk image, the "wood" texture was correctly identified. The regular pattern of windows were marked as texture in the hotel image. In the house image, the wood siding, trees, and part of the grass were labeled as texture (much of the grass was low contrast and labeled as "flat" texture). One of the bushes is correctly identified as having coarser texture than the other has. In the lighthouse image, the house sans window, fence, and tower were marked, as well as a low-frequency oriented pattern in the clouds.

Figure 5 shows the regions of interest that were found (the striped and plaid patterns here have no meaning but were chosen for maximum visibility). Most complex natural scenes had few interesting low-level attentional areas. In the lighthouse image, the life preserver is marked. In the hotel, curved or unusual angular windows are identified as attentional cues, as well as the top of the building. Both of these results are in agreement with psychophysical results showing that observers quickly identify curved or bent lines among straight lines (reviewed in Wolfe, 1998). The simpler desk scene yields more intuitive results, with each of the 3 objects labeled, as well as the phone cord.

Bottom-up attentional cues are outliers to the local distribution of features, and we have suggested that texture is the absence of such outliers. This definition captures some of the intuition that texture is homogeneous and statistical in nature. We presented a method for finding contrast and orientation outliers, and results both on localizing texture and on finding popout in natural images. For the simple desk image, the algorithm highlights salient regions that correspond to our notions of the important objects in the scene. On complicated natural scenes, its results are less intuitive; suggesting that search in natural scenes makes use of higher-level

processing such as grouping into objects. This result should not be terribly surprising, but serves as a useful check on simple low-level models of visual attention. The algorithm does a good job of identifying textured regions at a number of different scales, with the results perhaps more intuitive at finer scales.

## Acknowledgments

This work was partially supported by an NRC postdoctoral award at NASA Ames. Many thanks to David Marimont and Eric Saund for useful discussions.

## Footnotes

* Email: rruth@parc.xerox.com

## References

J. R. Bergen and M. S. Landy (1991), "Computational modeling of visual texture segmentation," *Computational Models of Visual Processing*, Landy and Movshon (eds.), pp. 252-271, MIT Press, Cambridge, MA.

J. Duncan and G. Humphreys (1989), "Visual search and stimulus similarity," *Psych. Review* **96**, pp. 433-458.

D. Forsyth, J. Malik, M. Fleck, H. Greenspan, T. Leung, S. Belongie, C. Carson, and C. Bregler (1996), "Finding pictures of objects in collections of images," *ECCV Workshop on Object Representation*, Cambridge.

F. A. A. Kingdom, D. Keeble, D., and B. Moulden (1995), "Sensitivity to orientation modulation in micropattern-based textures," *Vis. Res.* **35**, 1, pp. 79-91.

F. A. A. Kingdom and D. Keeble (1997), "The mechanism for scale invariance in orientation-defined textures." *Invest. Ophthal. and Vis. Sci. (Suppl.)* **38**, 4, p. 636.

T. K. Leung and J. Malik (1996), "Detecting, localizing, and grouping repeated scene elements from an image," *Proc. 4$^{th}$ European Conf. On Computer Vision*, **1064**, 1, pp. 546-555, Springer-Verlag, Cambridge.

R. Milanese, H. Wechsler, S. Gil, J. -M. Bost, and T. Pun (1993), "Integration of bottom-up and top-down cues for visual attention using non-linear relaxation," *Proc. IEEE CVPR*, pp. 781-785, IEEE Computer Society Press, Seattle.

R. Rosenholtz (1997), "Basic signal detection theory model does not explain search among heterogeneous distractors." *Invest. Ophthal. and Vis. Sci. (Suppl.)* **38**, 4, p. 687.

R. Rosenholtz (1998), "A simple saliency model explains a number of motion popout phenomena." *Invest. Ophthal. and Vis. Sci. (Suppl.)* **39**, 4, p. 629.

E. Saund (1998), "Scale and the Shape/Texture Continuum," Xerox Internal Technical Memorandum.

J. Shi and J. Malik (1998), "Self Inducing Relational Distance and its Application to Image Segmentation," *Proc. 5$^{th}$ European Conf. on Computer Vision*, Burkhardt and Neumann (eds.), **1406**, 1, pp. 528-543, Springer, Freiburg.

J. Vaisey and A. Gersho (1992), "Image compression with variable block size segmentation." *IEEE Trans. Signal Processing* **40**, 8, pp. 2040-2060.

J. M. Wolfe (1998), "Visual search: a review," *Attention*, H. Pashler (ed.), pp. 13-74, Psychology Press Ltd., Hove, East Sussex, UK.

C. S. Won and D. K. Park (1997), "Image block classification and variable block size segmentation using a model-fitting criterion," *Opt. Eng.* **36**, 8, pp. 2204-2209.

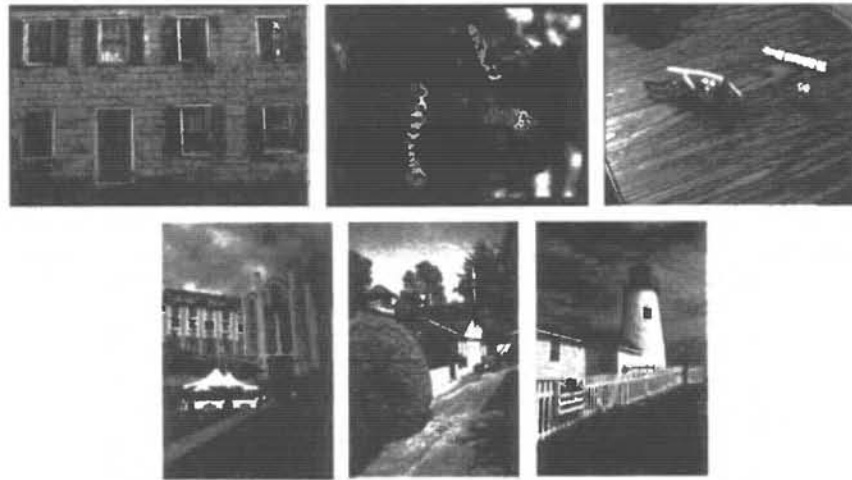

Figure 1: Original images.

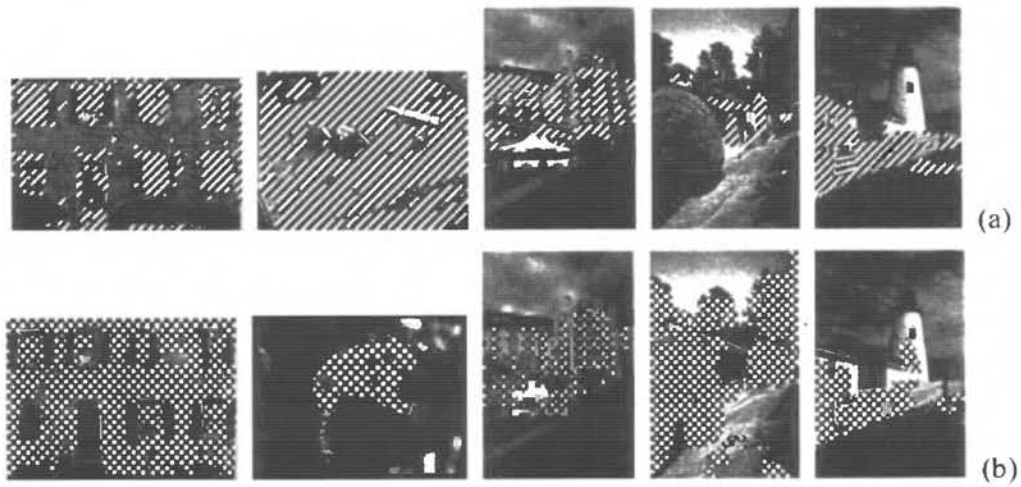

Figure 2: Fine-scale texture. (a) oriented texture, (b) homogeneous contrast texture.

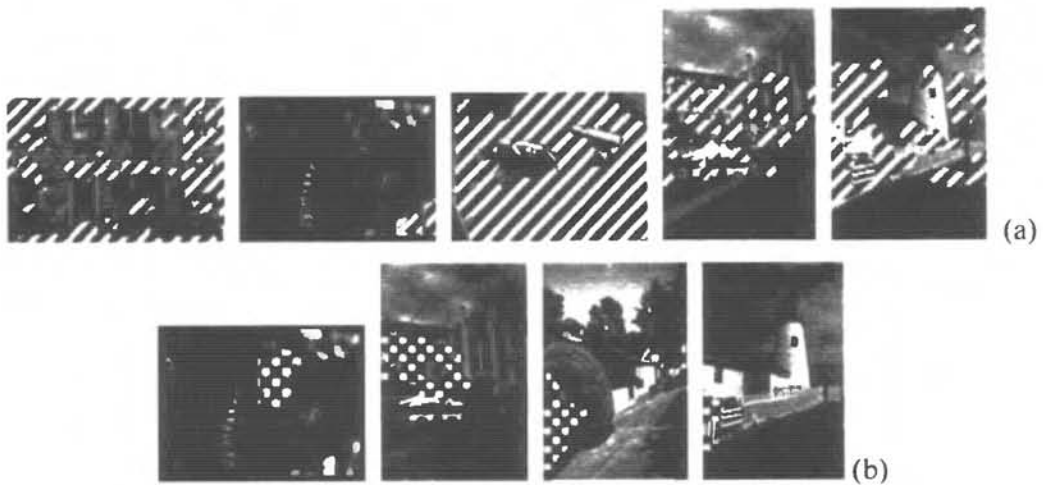

Figure 3: Medium-scale texture. (a) oriented texture, (b) homogeneous contrast texture.

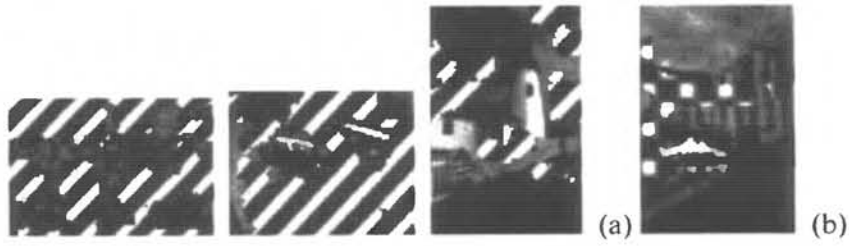

Figure 4: Coarse-scale texture. (a) oriented texture, (b) homogeneous contrast texture.

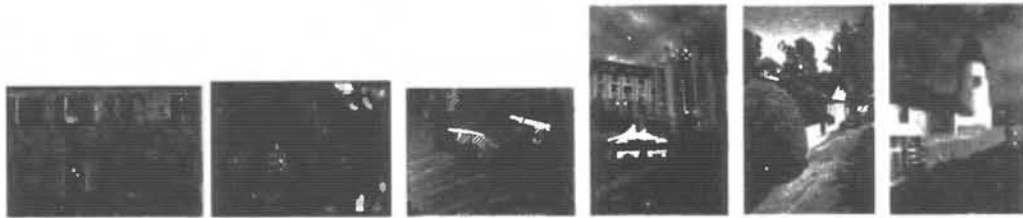

Figure 5: Regions of interest.